# Global Coordination of Local Linear Models

**Sam Roweis[†], Lawrence K. Saul[††], and Geoffrey E. Hinton[†]**
[†] Department of Computer Science, University of Toronto
[††] Department of Computer and Information Science, University of Pennsylvania

## Abstract

High dimensional data that lies on or near a low dimensional manifold can be described by a collection of local linear models. Such a description, however, does not provide a global parameterization of the manifold—arguably an important goal of unsupervised learning. In this paper, we show how to learn a collection of local linear models that solves this more difficult problem. Our local linear models are represented by a mixture of factor analyzers, and the "global coordination" of these models is achieved by adding a regularizing term to the standard maximum likelihood objective function. The regularizer breaks a degeneracy in the mixture model's parameter space, favoring models whose internal coordinate systems are aligned in a consistent way. As a result, the internal coordinates change smoothly and continuously as one traverses a connected path on the manifold—even when the path crosses the domains of many different local models. The regularizer takes the form of a Kullback-Leibler divergence and illustrates an unexpected application of variational methods: not to perform approximate inference in intractable probabilistic models, but to learn more useful internal representations in tractable ones.

## 1 Manifold Learning

Consider an ensemble of images, each of which contains a face against a neutral background. Each image can be represented by a point in the high dimensional vector space of pixel intensities. This representation, however, does not exploit the strong correlations between pixels of the same image, nor does it support many useful operations for reasoning about faces. If, for example, we select two images with faces in widely different locations and then average their pixel intensities, we do not obtain an image of a face at their average location. Images of faces lie on or near a low-dimensional, curved manifold, and we can represent them more usefully by the coordinates on this manifold than by pixel intensities. Using these "intrinsic coordinates", the average of two faces is another face with the average of their locations, poses and expressions.

To analyze and manipulate faces, it is helpful to imagine a "magic black box" with levers or dials corresponding to the intrinsic coordinates on this manifold. Given a setting of the levers and dials, the box generates an image of a face. Given an image of a face, the box deduces the appropriate setting of the levers and dials. In this paper, we describe a fairly general way to construct such a box automatically from an ensemble of high-dimensional vectors. We assume only that there exists an underlying manifold of low dimensionality and that the relationship between the raw data and the manifold coordinates is locally linear and smoothly varying. Thus our method applies not only to images of faces, but also to many other forms of highly distributed perceptual and scientific data (e.g., spectrograms of speech, robotic sensors, gene expression arrays, document collections).

## 2   Local Linear Models

The global structure of perceptual manifolds (such as images of faces) tends to be highly nonlinear. Fortunately, despite their complicated global structure, we can usually characterize these manifolds as locally linear. Thus, to a good approximation, they can be represented by collections of simpler models, each of which describes a locally linear neighborhood[3, 6, 8]. For unsupervised learning tasks, a probabilistic model that nicely captures this intuition is a *mixture of factor analyzers* (MFA)[5]. The model is used to describe high dimensional data that lies on or near a lower dimensional manifold. MFAs parameterize a joint distribution over observed and hidden variables:

$$P(\boldsymbol{x}, s, \boldsymbol{z}_s) = P(\boldsymbol{x}|s, \boldsymbol{z}_s)P(\boldsymbol{z}_s|s)P(s), \tag{1}$$

where the observed variable, $\boldsymbol{x} \in \mathcal{R}^D$, represents the high dimensional data; the discrete hidden variables, $s \in \{1, 2, \ldots, S\}$, indexes different neighborhoods on the manifold; and the continuous hidden variables, $\boldsymbol{z}_s \in \mathcal{R}^d$, represent low dimensional local coordinates. The model assumes that data is sampled from different neighborhoods on the manifold with prior probabilities $P(s) = p_s$, and that within each neighborhood, the data's local coordinates are normally distributed[1] as:

$$P(\boldsymbol{z}_s|s) = (2\pi)^{-d/2} \exp\left\{-\frac{1}{2}\boldsymbol{z}_s^\top \boldsymbol{z}_s\right\}. \tag{2}$$

Finally, the model assumes that the data's high and low dimensional coordinates are related by linear processes parameterized by centers $\boldsymbol{\mu}_s$, loading matrices $\Lambda_s$ and noise levels $\Psi_s$:

$$P(\boldsymbol{x}|s, \boldsymbol{z}_s) = |2\pi\Psi_s|^{-1/2} \exp\left\{-\frac{1}{2}[\boldsymbol{x} - \boldsymbol{\mu}_s - \Lambda_s\boldsymbol{z}_s]^\top \Psi_s^{-1} [\boldsymbol{x} - \boldsymbol{\mu}_s - \Lambda_s\boldsymbol{z}_s]\right\}. \tag{3}$$

The marginal data distribution, $P(\boldsymbol{x})$, is obtained by summing/integrating out the model's discrete and continuous latent variables. The result is a mixture of Gaussian distributions with parameterized covariance matrices of the form:

$$P(\boldsymbol{x}) = \sum_s p_s |2\pi(\Lambda_s\Lambda_s^\top + \Psi_s)|^{-1/2} \exp\left\{-\frac{1}{2}[\boldsymbol{x} - \boldsymbol{\mu}_s]^\top (\Lambda_s\Lambda_s^\top + \Psi_s)^{-1} [\boldsymbol{x} - \boldsymbol{\mu}_s]\right\}. \tag{4}$$

The learning problem for MFAs is to estimate the centers $\boldsymbol{\mu}_s$, transformations $\Lambda_s$, and noise levels $\Psi_s$ of these linear processes, as well as the prior probabilities $p_s$ of sampling data from different parts of the manifold. Parameter estimation in MFAs can be handled by an Expectation-Maximization (EM) algorithm[5] that attempts to maximize the log-probability, $\log P(\boldsymbol{x})$, averaged over training examples.

Note that the parameter space of this model exhibits an invariance: taking $\Lambda_s \to \Lambda_s R_s$, where $R_s$ are $d \times d$ orthogonal matrices ($R_s R_s^\top = I$), does not change the marginal distribution, $P(\boldsymbol{x})$. The transformations $\Lambda_s \to \Lambda_s R_s$ correspond to arbitrary rotations and reflections of the local coordinates in each linear model. The objective function for the EM algorithm is unchanged by these transformations. Thus, maximum likelihood estimation in MFAs does not favor any particular alignment; instead, it produces models whose internal representations change unpredictably as one traverses connected paths on the manifold. Can we encourage models whose local coordinate systems are aligned in a consistent way?

## 3   Global Coordination

Suppose the data lie near a smooth manifold with a locally flat (developable) structure. Then there exist a single set of "global coordinates" $\boldsymbol{g}$ which parametrize the manifold

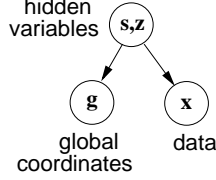

hidden
variables
global
coordinates
data

**Figure 1:** Graphical model for globally coordinated MFAs. Although global coordinates $\boldsymbol{g}$ are unobserved, they affect the learning through a regularization term. After learning, inferences about the global variables are made by computing posterior distributions, $P(\boldsymbol{g}|\boldsymbol{x})$. Likewise, data can easily be generated by sampling from the conditional distribution, $P(\boldsymbol{x}|\boldsymbol{g})$. All these operations are particularly tractable due to the conditional independencies of the model.

everywhere. Furthermore, to a good approximation, these global coordinates can be related to the local coordinates of different neighborhoods (in their region of validity) by linear[2] transformations:

$$g(s, \boldsymbol{z}_s) \; = \; A_s \boldsymbol{z}_s + \boldsymbol{\kappa}_s. \tag{5}$$

What does it mean to say that the coordinates $g(s, \boldsymbol{z}_s)$ provide a global parameterization of the manifold? Intuitively, if a data point belongs to overlapping neighborhoods, then the global coordinates computed from their local coordinate systems, given by eq. (5), should agree. We can formalize this "global coordination" of different local models by treating the coordinates $\boldsymbol{g}$ as unobserved variables and incorporating them into the probabilistic model:

$$P(\boldsymbol{g}|s, \boldsymbol{z}_s) \; = \delta(\boldsymbol{g} - A_s \boldsymbol{z}_s - \boldsymbol{\kappa}_s), \tag{6}$$

(Here we posit a deterministic relationship between local and global coordinates, although it is possible to add noise to this mapping as well.) The globally coordinated MFA is represented by the graphical model in Fig. 1. We can appeal to its conditional independencies to make other useful inferences. In particular:

$$P(\boldsymbol{g}|\boldsymbol{x}_n, s) \;\; = \;\; \int d\boldsymbol{z}_s \; P(\boldsymbol{g}|s, \boldsymbol{z}_s) P(\boldsymbol{z}_s|s, \boldsymbol{x}_n) \tag{7}$$

$$P(\boldsymbol{g}|\boldsymbol{x}_n) \;\; = \;\; \sum_s P(s|\boldsymbol{x}_n) P(\boldsymbol{g}|\boldsymbol{x}_n, s). \tag{8}$$

Now, if two or more mixture components—say, $s_1$ and $s_2$—explain a data point $\boldsymbol{x}_n$ with non-negligible probability, then the posterior distributions for the global coordinates of this data point, as induced by eq. (8), should be nearly identical: that is, $P(\boldsymbol{g}|\boldsymbol{x}_n, s_1) \approx P(\boldsymbol{g}|\boldsymbol{x}_n, s_2)$. To enforce this criterion of agreement, we need to penalize models whose posterior distributions $P(\boldsymbol{g}|\boldsymbol{x}_n)$ given by eq. (8) are *multimodal*, since multiple modes only arise when different mixture components give rise to inconsistent global coordinates. While directly penalizing multimodality of $P(\boldsymbol{g}|\boldsymbol{x}_n)$ is difficult, a penalty which encourages consistency *can* be easily incorporated into the learning algorithm. We introduce a family of unimodal distributions over both $\boldsymbol{g}$ and $s$, and encourage the true posteriors, $P(\boldsymbol{g}, s|\boldsymbol{x}_n)$, to be close to some member, $Q(\boldsymbol{g}, s|\boldsymbol{x}_n)$, of this family.

Developing this idea further, we introduce a new objective function for unsupervised learning in MFAs. The new objective function incorporates a regularizer to encourage the global consistency of local models:

$$\Phi = \sum_n \log P(\boldsymbol{x}_n) - \lambda \sum_{ns} \int d\boldsymbol{g} \; Q(\boldsymbol{g}, s|\boldsymbol{x}_n) \log \left[ \frac{Q(\boldsymbol{g}, s|\boldsymbol{x}_n)}{P(\boldsymbol{g}, s|\boldsymbol{x}_n)} \right], \tag{9}$$

The first term in this objective function computes the log-probability of the data. The second term computes a sum of Kullback-Leibler (KL) divergences; these are designed to

penalize MFAs whose posterior distributions over global coordinates are not unimodal. The twin goals of density estimation and manifold learning in MFAs are pursued by attempting to balance these terms in the objective function. The factor $\lambda$ controls the tradeoff between density modeling and global coordination: as $\lambda \to 0$ only strict invariances (which do not affect likelihood) are exploited in order to achieve submodel agreement. In what follows we have set $\lambda = 1$ arbitrarily; further optimization is possible.

The most convenient way to parameterize the family of unimodal distributions is a factorized form involving a Gaussian density and a multinomial:

$$Q(\boldsymbol{g}, s|\boldsymbol{x}_n) = Q(\boldsymbol{g}|\boldsymbol{x}_n)Q(s|\boldsymbol{x}_n); \quad Q(\boldsymbol{g}|\boldsymbol{x}_n) \sim \mathcal{N}(\boldsymbol{g}_n, \Sigma_n); \qquad Q(s|\boldsymbol{x}_n) = q_{ns} \quad (10)$$

Note that the distribution $Q(\boldsymbol{g}, s|\boldsymbol{x}_n)$ in eq. (10) factorizes over $s$ and $\boldsymbol{g}$, implying that—according to this family of models—the global coordinate $\boldsymbol{g}$ is independent of the mixture component $s$ given the data point $\boldsymbol{x}_n$. Also, $Q(\boldsymbol{g}|\boldsymbol{x}_n)$ is Gaussian, and thus unimodal. These are exactly the constraints we wish to impose on the posterior $P(\boldsymbol{g}, s|\boldsymbol{x}_n)$. At each iteration of learning, the means $\boldsymbol{g}_n$, covariance matrices $\Sigma_n$, and mixture weights $q_{ns}$ are determined separately for each data point, $\boldsymbol{x}_n$ so as to maximize the objective function in eq. (9): this amounts to computing the unimodal distributions, $Q(\boldsymbol{g}, s|\boldsymbol{x}_n)$, best matched to the true posterior distributions, $P(\boldsymbol{g}, s|\boldsymbol{x}_n)$.

## 4  Learning Algorithm

Latent variable models are traditionally estimated by maximum likelihood or Bayesian methods whose objective functions do not reward the interpretability of their internal representations. Note how the goal of developing more useful internal representations has changed the learning problem in a fundamental way. Now we have additional "coordination" parameters–the offsets $\boldsymbol{\kappa}_s$ and weights $A_s$–that must also be learned from examples. We also have auxiliary parameters for each data point–the means $\boldsymbol{g}_n$, covariance matrices $\Sigma_n$, and mixture weights $q_{ns}$—that determine the target distributions, $Q(\boldsymbol{g}, s|\boldsymbol{x}_n)$. All these parameters, as well as the MFA model parameters $\{p_s, \Lambda_s, \boldsymbol{\mu}_s, \Psi_s\}$, must be chosen to "stitch together" the local coordinates systems in a smooth way and to learn internal representations easily coordinated by the local-to-global mapping in eq. (6).

Optimization of the objective function in eq. (9) is reminiscent of so-called "variational" methods for approximate learning[7]. In these methods, an approximation to an exact (but intractable) posterior distribution is fitted by minimizing a KL divergence between the two distributions. The auxiliary parameters of the approximating distribution are known as variational parameters. Our objective function illustrates an unexpected application of such variational methods: not to perform approximate inference in intractable probabilistic models, but to learn more useful internal representations in tractable ones. We introduce the unimodal and factorized distributions $Q(\boldsymbol{g}, s|\boldsymbol{x}_n)$ to regularize the multimodal distributions $P(\boldsymbol{g}, s|\boldsymbol{x}_n)$. Penalizing the KL divergence between these distributions lifts a degeneracy in the model's parameter space and favors local linear models that can be globally aligned.

### 4.1  Computing and optimizing the objective function

Evaluating the objective function in eq. (9) requires a sum and integral over the latent variables of the model. These operations are simplified by rewriting the objective function as:

$$\Phi = \sum_{ns} \int d\boldsymbol{g} \ Q(\boldsymbol{g}, s|\boldsymbol{x}_n) \left[ -\log Q(\boldsymbol{g}, s|\boldsymbol{x}_n) + \log P(\boldsymbol{x}_n, \boldsymbol{g}, s) \right]. \qquad (11)$$

The factored form of the distributions $Q(\boldsymbol{g}, s|\boldsymbol{x}_n)$ makes it straightforward to perform the required sums and integrals. The final result is a simple form in terms of entropies $\mathcal{S}_{ns}$ and

energies $\mathcal{E}_{ns}$ associated with the $n$th data point:

$$\Phi = \sum_{ns} q_{ns} \left( \mathcal{S}_{ns} - \mathcal{E}_{ns} \right), \tag{12}$$

$$\mathcal{S}_{ns} = \frac{1}{2} \log |\Sigma_n| - \log q_{ns} + \frac{d}{2} \log(2\pi), \tag{13}$$

$$\mathcal{E}_{ns} = \frac{1}{2} g_{ns}^\top V_s g_{ns} + \frac{1}{2} x_{ns}^\top \Psi_s^{-1} x_{ns} - g_{ns}^\top A_s^{-\top} \Lambda_s^\top \Psi_s^{-1} x_{ns} + \frac{1}{2} \mathrm{Tr}[\Sigma_n V_s]$$
$$+ \frac{1}{2} \log |\Psi_s| + \log |A_s| - \log p_s + \frac{D+d}{2} \log(2\pi), \tag{14}$$

where we have introduced simplifying notation for the vector differences $x_{ns} = (x_n - \mu_s)$ and $g_{ns} = (g_n - \kappa_s)$ and the local precision matrices $V_s = A_s^{-1} \left( I + \Lambda_s^\top \Psi_s^{-1} \Lambda_s \right) A_s^{-1}$.

Iteratively maximizing the objective function by coordinate ascent now leads to a learning algorithm of the same general style as EM.

## 4.2 E-step

Maximizing the objective function, eq. (9), with respect to the regularizing parameters $\{g_n, \Sigma_n, q_{ns}\}$ (and subject to the constraint $\sum_s q_{ns} = 1$) leads to the fixed point equations:

$$\Sigma_n = \left[ \sum_s q_{ns} V_s \right]^{-1} \qquad g_n = \Sigma_n \left[ \sum_s q_{ns} m_{ns} \right] \qquad q_{ns} = \frac{e^{-\mathcal{E}_{ns}}}{\sum_{s'} e^{-\mathcal{E}_{ns'}}}. \tag{15}$$

where $m_{ns} = V_s \kappa_s + A_s^{-\top} \Lambda_s^\top \Psi_s^{-1} x_{ns}$. These equations can be solved by iteration with initialization $q_{ns} = p_s$. Notice that $V_s$ and $m_{ns}$ only need to be computed once before iterating the fixed point equations. The objective function is completely invariant to translation and rescaling of $g_n$ and $\kappa_s$ (since $A_s$, $\kappa_s$ and $g_n$ appear only in the form $(g_n - \kappa_s) A_s^{-\top}$). To remove this degeneracy, after solving the equations above we further constrain the global coordinates to have mean zero and unit variance in each direction. These constraints are enforced without changing the value of the objective function by simply translating the offsets $\kappa_s$ and rescaling the diagonal matrices $A_s$.

## 4.3 M-step

The M-step consists of maximizing the objective function, eq. (9), with respect to the generative model parameters. Let us denote the updated parameter estimates by $\{\tilde{p}_s, \tilde{\kappa}_s, \tilde{\mu}_s, \tilde{\Lambda}_s, \tilde{\Psi}_s, \tilde{A}_s\}$. Letting $q_s = \sum_n q_{ns}$, the M-step updates for the first three of these are:

$$\tilde{p}_s \leftarrow q_s / \sum_{s'} q_{s'} \qquad \tilde{\kappa}_s \leftarrow q_s^{-1} \sum_n q_{ns} g_n \qquad \tilde{\mu}_s \leftarrow q_s^{-1} \sum_n q_{ns} x_n. \tag{16}$$

The remaining updates, to be performed in the order shown, are given in terms of updated difference vectors $\tilde{x}_{ns} = x_n - \tilde{\mu}_s$, $\tilde{g}_{ns} = g_n - \tilde{\kappa}_s$, the correlations $C_s = \sum_n q_{ns} \tilde{x}_{ns} \tilde{g}_{ns}^\top$, and the variances $G_s = \sum_n q_{ns} [\Sigma_n + \tilde{g}_{ns} \tilde{g}_{ns}^\top]$.

$$\tilde{\Lambda}_s \leftarrow C_s G_s^{-1} A_s, \tag{17}$$

$$\left[ \tilde{\Psi}_s \right]_i \leftarrow q_s^{-1} \sum_n q_{ns} \left\{ \left[ \tilde{x}_{ns} - \tilde{\Lambda}_s A_s^{-1} \tilde{g}_{ns} \right]_i^2 + \left[ \tilde{\Lambda}_s A_s^{-1} \Sigma_n A_s^{-\top} \tilde{\Lambda}_s^\top \right]_i \right\} \tag{18}$$

$$\tilde{A}_s^{-1} \leftarrow (I + \Lambda_s^\top \Psi_s^{-1} \Lambda_s)^{-1} \left\{ A_s^\top q_s + \Lambda_s^\top \tilde{\Psi}_s^{-1} C_s \right\} G_s^{-1} \tag{19}$$

At the optimum, the coordination weights $A_s$ satisfy an algebraic Riccati equation which can be solved by iterating the update shown above. (Such equations can also be solved by much more sophisticated methods well known in the engineering community. Most approaches involve inverting the previous value of $A_s$ which may be expensive for full matrices but is fast in our diagonal implementation.)

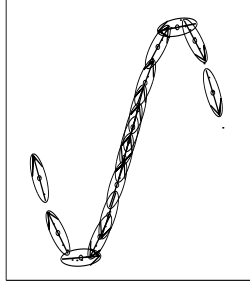 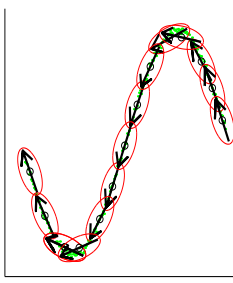

**Figure 2:** Global coordination of local linear models. **(left)** A model trained using maximum likelihood, with the arrows indicating the direction of increase for each factor analyzer's local coordinate system. **(right)** A coordinated model; arrows indicate the direction in the data space corresponding to increasing the global coordinate $g$ as inferred by the algorithm. The ellipses show the one standard deviation contour of the density of each analyzer.

## 5 Experiments

We have tested our model on simple synthetic manifolds whose structure is known as well as on collections of images of handwritten digits and faces. Figure 2 illustrates the basic concept of coordination, as achieved by our learning rule. In the coordinated model, the global coordinate always points in the same direction along the data manifold, as defined by the composition of the transformations $\Lambda_s$ and $A_s$. In the model trained with maximum likelihood, the density is well captured but each local latent variable has a random orientation along the manifold.

We also applied the algorithm to collections of images of handwritten digits and of faces. The representation of $x$ was an unprocessed vector of raw 8-bit grayscale pixel intensities for each image (of dimensionality 256 for the $16 \times 16$ digits and 560 for the $28 \times 20$ faces.) The MFAs had 64 local models and the global coordinates were two dimensional. After training, the coordinated MFAs had learned a smooth, continuous mapping from the plane to images of digits or of faces. This allows us both to infer a two-dimensional location given any image by computing $P(g|x)$ and to generate new images from any point in the plane by computing $P(x|g)$. (Precisely what we wanted from the magic box.) In general, both of these conditional distributions have the form of a mixture of Gaussians. Figure 3 shows the inferred global coordinates $g_n$ (i.e. the means of the unimodal distributions $Q(g|x_n)$) of the training points after the last iteration of training as well as examples of new images from the generative model, created by evaluating the mean of $P(x|g)$ along straight line paths in the global coordinate space. In the case of digits, it seems as though our models have captured tilt/shape and identity and represented them as the two axes of the $g$ space; in the case of the faces the axes seem to capture pose and expression. (For the faces, the final $g$ space was rotated by hand to align interpretable directions with the coordinate axes.)

As with all EM algorithms, the coordinated MFA learning procedure is susceptible to local optima. Crucial to the success of our experiments is a good initialization, which was provided by the Locally Linear Embedding algorithm[9]. We clamped $g_n$ equal to the embedding coordinate provided by LLE and $\Sigma_n$ to a small value and trained until convergence (typically 30-100 iterations). Then we proceeded with training using the full EM equations to update $g_n$, again until convergence (usually 5-10 more iterations). Note, however, that LLE and other embedding algorithms such as Isomap[10] are themselves unsupervised, so the overall procedure, including this initial phase, is still unsupervised.

## 6 Discussion

Mixture models provide a simple way to approximate the density of high dimensional data that lies on or near a low dimensional manifold. However, their hidden representations do not make explicit the relationship between dissimilar data vectors. In this paper, we have shown how to learn global coordinates that can act as an encapsulating interface, so that other parts of a learning system do not need to interact with the individual components of a mixture. This should improve generalization as well as facilitate the propagation and exchange of information when these models are incorporated into a larger (perhaps

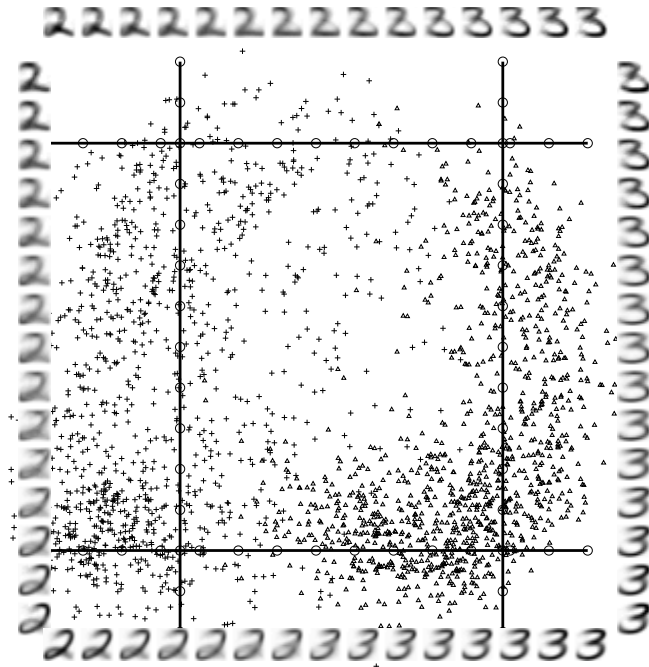

**Figure 3:** Automatically constructed two dimensional global parameterizations of manifolds of digits and faces. Each plot shows the global coordinate space discovered by the unsupervised algorithm; points indicate the inferred means $g_n$ for each training item at the end of learning. The image stacks on the borders are not from the training set but are generated from the model itself and represent the mean of the predictive distribution $P(x|g)$ at the corresponding open circles (sampled along the straight lines in the global space).

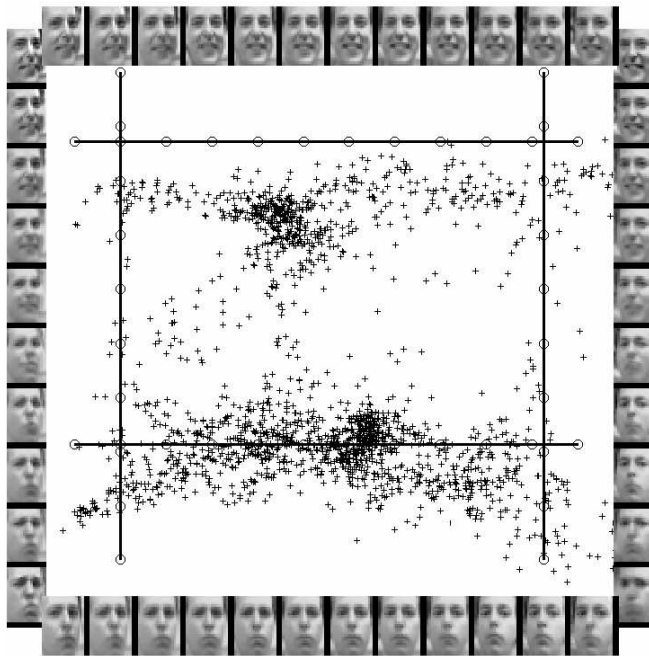

The models provide both a two degree-of-freedom generator for complex images via $P(x|g)$ as well as a pose/slant recognition system via $P(g|x)$.

For the handwritten digits, the training set consisted of 1100 examples of the digit "2" (shown as crosses above) mixed with 1100 examples of "3"s (shown as triangles). The digits are from the NIST dataset, digitized at 16x16 pixels. For the faces, we used 2000 images of a single person with various poses and expressions taken from consecutive frames of a video digitized at 20x20 pixels. Brendan Frey kindly provided the face data.

hierarchical) architecture for probabilistic reasoning.

Two variants of our purely unsupervised proposal are possible. The first is to use an embedding algorithm (such as LLE or Isomap) not only as an initialization step but to provide clamped values for the global coordinates. While this supervised approach may work in practice, unsupervised coordination makes clear the objective function that is being opti-

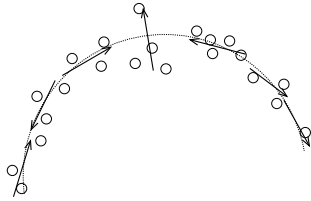

**Figure 4:** A situation in which an un-coordinated mixture model–trained to do density estimation–cannot be "post-coordinated". Noise has caused one of the local density models to orient orthogonal to the manifold. In globally coordinated learning, there is an additional pressure to align with neighbouring models which would force the local model to lie in the correct subspace.

mized, which unifies the goals of manifold learning and density estimation. Another variant is to train an unsupervised mixture model (such as a MFA) using a traditional maximum likelihood objective function and then to "post-coordinate" its parameters by applying local reflections/rotations and translations to create global coordinates. As illustrated in figure 4, however, this two-step procedure can go awry because of noise in the original training set. When both density estimation and coordination are optimized simultaneously there is extra pressure for local experts to fit the global structure of the manifold.

Our work can be viewed as a synthesis of two long lines of research in unsupervised learning. In the first are efforts at learning the global structure of nonlinear manifolds [1, 4, 9, 10]; in the second are efforts at developing probabilistic graphical models for reasoning under uncertainty[5, 6, 7]. Our work proposes to model the global coordinates on manifolds as latent variables, thus attempting to combine the representational advantages of both frameworks. It differs from embedding by providing a fully probabilistic model valid away from the training set, and from work in generative topographic mapping[2] by not requiring a uniform discretized gridding of the latent space. Moreover, by extending the usefulness of mixture models,it further develops an architecture that has already proved quite powerful and enormously popular in applications of statistical learning.

### Acknowledgements

We thank Mike Revow for sharing his unpublished work (at the University of Toronto) on coordinating mixtures, and Zoubin Ghahramani, Peter Dayan, Jakob Verbeek and two anonymous reviewers for helpful comments and corrections.

## Footnotes

[1]Although in principle each neighborhood could have a different prior on its local coordinates, without loss of generality we have made the standard assumption that $P(\boldsymbol{z}_s|s)$ is the same for all settings of $s$ and absorbed the shape of each local Gaussian model into the matrices $\Lambda_s$.

[2]Without loss of generality, the matrices $A_s$ can be taken to be symmetric and positive-definite, by exploiting the polar factorization and absorbing reflection and rotation into the local coordinate systems. (In practice, though, it may be easier to optimize the objective function without constraining the matrices to be of this form.) In the experiments reported below, we have further restricted them to be diagonal. Together, then, the coordination matrices $A_s$ and vectors $\boldsymbol{\kappa}_s$ account for an axis-aligned scaling and uniform translation between the global and local coordinate systems.

## References

[1] D. Beymer & T. Poggio. Image representations for visual learning. pringer*Science* **272** (1996).

[2] C. Bishop, M. Svensen, and C. Williams. GTM: The generative topographic mapping. *Neural Computation* **10** (1998).

[3] C. Bregler & S. Omohundro. Nonlinear image interpolation using manifold learning. *Advances in Neural Information Processing Systems 7* (1995).

[4] D. DeMers & G.W. Cottrell. Nonlinear dimensionality reduction. *Advances in Neural Information Processing Systems 5* (1993).

[5] Ghahramani, Z. and Hinton, G. The EM algorithm for mixtures of factor analyzers. *University of Toronto Technical Report* CRG-TR-96-1 (1996).

[6] Hinton, G., Dayan, P., and Revow, M. Modeling the manifolds of images of handwritten digits. *IEEE Transactions on Neural Networks* **8** (1997).

[7] M. Jordan, Z. Ghahramani, T. Jaakkola, and L. Saul. An introduction to variational methods for graphical models. *Machine Learning* **37**(2) (1999).

[8] N. Kambhatla and T. K. Leen. Dimension reduction by local principal component analysis. *Neural Computation* **9** (1997).

[9] S. T. Roweis & L. K. Saul. Nonlinear dimensionality reduction by locally linear embedding. *Science* **290** (2000).

[10] J. B. Tenenbaum, V. de Silva, and J. C. Langford. A global geometric framework for nonlinear dimensionality reduction. *Science* **290** (2000).
